# Balancing Multiple Sources of Reward in Reinforcement Learning

**Christian R. Shelton**
Artificial Intelligence Lab
Massachusetts Institute of Technology
Cambridge, MA 02139
*cshelton@ai.mit.edu*

## Abstract

For many problems which would be natural for reinforcement learning, the reward signal is not a single scalar value but has multiple scalar components. Examples of such problems include agents with multiple goals and agents with multiple users. Creating a single reward value by combining the multiple components can throw away vital information and can lead to incorrect solutions. We describe the multiple reward source problem and discuss the problems with applying traditional reinforcement learning. We then present an new algorithm for finding a solution and results on simulated environments.

## 1 Introduction

In the traditional reinforcement learning framework, the learning agent is given a single scalar value of reward at each time step. The goal is for the agent to optimize the sum of these rewards over time (the return). For many applications, there is more information available.

Consider the case of a home entertainment system designed to sense which residents are currently in the room and automatically select a television program to suit their tastes. We might construct the reward signal to be the total number of people paying attention to the system. However, a reward signal of 2 ignores important information about which two users are watching. The users of the system change as people leave and enter the room. We could, in theory, learn the relationship among the users present, who is watching, and the reward. In general, it is better to use the domain knowledge we have instead of requiring the system to learn it. We know which users are contributing to the reward and that only present users can contribute.

In other cases, the multiple sources aren't users, but goals. For elevator scheduling we might be trading off people serviced per minute against average waiting time. For financial portfolio managing, we might be weighing profit against risk. In these cases, we may wish to change the weighting over time. In order to keep from having to relearn the solution from scratch each time the weighting is changed, we need to keep track of which rewards to attribute to which goals.

There is a separate difficulty if the rewards are not designed functions of the state but

rather are given by other agents or people in the environment. Consider the case of the entertainment system above but where every resident has a dial by which they can give the system feedback or reward. The rewards are incomparable. One user may decide to reward the system with values twice as large as those of another which should not result in that user having twice the control over the entertainment. This isn't limited to scalings but also includes any other monotonic transforms of the returns. If the users of the system know they are training it, they will employ all kinds of reward strategies to try to steer the system to the desired behavior [2]. By keeping track of the sources of the rewards, we will derive an algorithm to overcome these difficulties.

## 1.1 Related Work

The work presented here is related to recent work on multiagent reinforcement learning [1, 4, 5, 7] in that multiple rewards signals are present and game theory provides a solution. This work is different in that it attacking a simpler problem where the computation is consolidated on a single agent. Work in multiple goals (see [3, 8] as examples) is also related but assumes either that the returns of the goals are to be linearly combined for an overall value function or that only one goal is to be solved at a time.

## 1.2 Problem Setup

We will be working with partially observable environments with discrete actions and discrete observations. We make no assumptions about the world model and thus do not use belief states. $x(t)$ and $a(t)$ are the observation and action, respectively, at time $t$. We consider only reactive policies (although the observations could be expanded to include history). $\pi(x, a)$ is the policy or probability the agent will take action $a$ when observing $x$. At each time step, the agent receives a set of rewards (one for each source in the environment), $r_s(t)$ is the reward at time $t$ from source $s$. We use the average reward formulation and so $R_s^\pi = \lim_{n \to \infty} \frac{1}{n} E\left[r_s(1) + r_s(2) + \cdots + r_s(n) | \pi\right]$ is the expected return from source $s$ for following policy $\pi$. It is this return that we want to maximize for each source.

We will also assume that the algorithm knows the set of sources present at each time step. Sources which are not present provide a constant reward, regardless of the state or action, which we will assume to be zero. All sums over sources will be assumed to be taken over only the present sources.

The goal is to produce an algorithm that will produce a policy based on previous experience and the sources present. The agent's experience will take the form of prior interactions with the world. Each experience is a sequence of observations, action, and reward triplets for a particular run of a particular policy.

# 2 Balancing Multiple Rewards

## 2.1 Policy Votes

If rewards are not directly comparable, we need to find a property of the sources which is comparable and a metric to optimize. We begin by noting that we want to limit the amount of control any given source has over the behavior of the agent. To that end, we construct the policy as the average of a set of votes, one for each source present. The votes for a source must sum to 1 and must all be non-negative (thus giving each source an equal "say" in the agent's policy). We will first consider restricting the rewards from a given source to only affect the votes for that source.

The form for the policy is therefore

$$\pi(x, a) = \frac{\sum_s \alpha_s(x) v_s(x, a)}{\sum_s \alpha_s(x)} \qquad (1)$$

where for each present source $s$, $\sum_x \alpha_s(x) = 1$, $\alpha_s(x) \geq 0$ for all $x$, $\sum_a v_s(x, a) = 1$ for all $x$, and $v_s(x, a) \geq 0$ for all $x$ and $a$. We have broken apart the vote from a source into two parts, $\alpha$ and $v$. $\alpha_s(x)$ is how much effort source $s$ is putting into affecting the policy for observation $x$. $v_s(x, a)$ is the vote by source $s$ for the policy for observation $x$. Mathematically this is the same as constructing a single vote ($v'_s(x, a) = \alpha_s(x) v_s(x, a)$), but we find $\alpha$ and $v$ to be more interpretable.

We have constrained the total effort and vote any one source can apply. Unfortunately, these votes are not quite the correct parameters for our policy. They are not invariant to the other sources present. To illustrate this consider the example of a single state with two actions, two sources, and a learning agent with the voting method from above. If $s_1$ prefers only $a_1$ and $s_2$ likes an equal mix of $a_1$ and $a_2$, the agent will learn a vote of (1,0) for $s_1$ and $s_2$ can reward the agent to cause it to learn a vote of (0,1) for $s_2$ resulting in a policy of (0.5,0.5). Whether this is the correct final policy depends on the problem definition. However, the real problem arises when we consider what happens if $s_1$ is removed. The policy reverts to $(0, 1)$ which is far from $s_2$'s (the only present source's) desired $(0.5, 0.5)$ Clearly, the learned votes for $s_2$ are meaningless when $s_1$ is not present.

Thus, while the voting scheme does limit the control each present source has over the agent, it does not provide a description of the source's preferences which would allow for the removal or addition (or reweighting) of sources.

## 2.2 Returns as Preferences

While rewards (or returns) are not comparable across sources, they are comparable within a source. In particular, we know that if $R_s^{\pi_1} > R_s^{\pi_2}$ that source $s$ prefers policy $\pi_1$ to policy $\pi_2$. We do not know how to weigh that preference against a different source's preference so an explicit tradeoff is still impossible, but we can limit (using the voting scheme of equation 1) how much one source's preference can override another source's preference.

We allow a source's preference for a change to prevail in as much as its votes are sufficient to affect the change in the presences of the other sources' votes. We have a type of a general-sum game (letting the sources be the players of game theory jargon). The value to source $s'$ of the set of all sources' votes is $R_{s'}^{\pi}$, where $\pi$ is the function of the votes defined in equation 1. Each source $s'$ would like to set its particular votes, $\alpha_{s'}(x)$ and $v'_s(x, a)$ to maximize its value (or return). Our algorithm will set each source's vote in this way thus insuring that no source could do better by "lying" about its true reward function.

In game theory, a "solution" to such a game is called a Nash Equilibrium [6], a point at which each player (source) is playing (voting) its best response to the other players. At a Nash Equilibrium, no single player can change its play and achieve a gain. Because the votes are real-valued, we are looking for the equilibrium of a continuous game. We will derive a fictitious play algorithm to find an equilibrium for this game.

## 3 Multiple Reward Source Algorithm

### 3.1 Return Parameterization

In order to apply the ideas of the previous section, we must find a method for finding a Nash Equilibrium. To do that, we will pick a parametric form for $\hat{R}_s^{\pi}$ (the estimate of the

return): linear in the KL-divergence between a target vote and $\pi$. Letting $a_s$, $b_s$, $\beta_s(x)$, and $p_s(x, a)$ be the parameters of $\hat{R}_s^\pi$,

$$\hat{R}_s^\pi = a_s \sum_x \beta_s(x) \sum_a p_s(x, a) \log \frac{p_s(x, a)}{\pi(x, a)} + b_s \tag{2}$$

where $a_s \geq 0$, $\beta_s(x) \geq 0$, $\sum_x \beta_s(x) = 1$, $p_s(x, a) \geq 0$, and $\sum_a p_s(x, a) = 1$. Just as $\alpha_s(x)$ was the amount of vote source $s$ was putting towards the policy for observation $x$, $\beta_s(x)$ is the importance for source $s$ of the policy for observation $x$. And, while $v_s(x, a)$ was the policy vote for observation $x$ for source $s$, $p_s(x, a)$ is the preferred policy for observation $x$ for source $s$. The constants $a_s$ and $b_s$ allow for scaling and translation of the return.

If we let $p'_s(x, a) = a_s \beta_s(x) p_s(x, a)$, then, given experiences of different policies and their empirical returns, we can estimate $p'_s(x, a)$ using linear least-squares. Imposing the constraints just involves finding the normal least-squares fit with the constraint that all $p'_s(x, a)$ be non-negative. From $p'_s(x, a)$ we can calculate $a_s = \sum_{x,a} p'_s(x, a)$, $\beta_s(x) = \frac{1}{a_s} \sum_a p'_s(x, a)$ and $p_s(x, a) = \frac{p'_s(x, a)}{\sum_{a'} p'_s(x, a')}$. We now have a method for solving for $\hat{R}_s^\pi$ given experience. We now need to find a way to compute the agent's policy.

## 3.2   Best Response Algorithm

To produce an algorithm for finding a Nash Equilibrium, let us first start by deriving an algorithm for finding the best response for source $s$ to a set of votes. We need to find the set of $\alpha_s(x)$ and $v_s(x, a)$ that satisfy the constraints on the votes and maximize equation 2 which is the same as minimizing

$$\sum_x \beta_s(x) \sum_a p_s(x, a) \log \frac{\sum_{s'} \alpha_{s'}(x) v_{s'}(x)}{\sum_{s'} \alpha_{s'}(x)} \tag{3}$$

over $\alpha_s(x)$ and $v_s(x, a)$ for given $s$ because the other terms depend on neither $\alpha_s(x)$ nor $v_s(x, a)$.

To minimize equation 3, let's first fix the $\alpha$-values and optimize $v_s(x, a)$. We will ignore the non-negative constraints on $v_s(x, a)$ and just impose the constraint that $\sum_a v_s(x, a) = 1$. The solution, whose derivation is simple and omitted due to space, is

$$v_s(x, a) = \frac{\sum_{s' \neq s} \alpha_{s'}(x)(p_s(x, a) - v_{s'}(x, a))}{\alpha_s(x)} . \tag{4}$$

We impose the non-negative constraints by setting to zero any $v_s(x, a)$ which are negative and renormalizing.

Unfortunately, we have not been able to find such a nice solution for $\alpha_s(x)$. Instead, we use gradient descent to optimize equation 3 yielding

$$\Delta \alpha_s(x) \propto \frac{\beta_s(x)}{\sum_{s'} \alpha_{s'}(x)} \sum_a p_s(x, a) \frac{\sum_{s' \neq s} \alpha_{s'}(x)(v_s(x, a) - v_{s'}(x, a))}{\sum_{s'} \alpha_{s'}(x) v_{s'}(x, a)} . \tag{5}$$

We constrain the gradient to fit the constraints.

We can find the best response for source $s$ by iterating between the two steps above. First we initialize $\alpha_s(x) = \beta_s(x)$ for all $x$. We then solve for a new set of $v_s(x, a)$ with equation 4. Using those $v$-values, we take a step in the direction of the gradient of $\alpha_s(x)$ with equation 5. We keep repeating until the solution converges (reducing the step size each iteration) which usually only takes a few tens of steps.

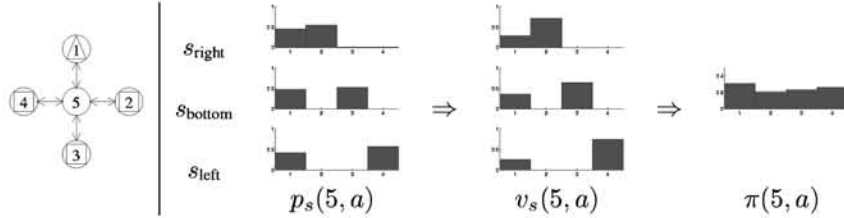

Figure 1: Load-unload problem: The right is the state diagram. Cargo is loaded in state 1. Delivery to a boxed state results in reward from the source associated with that state. The left is the solution found. For state 5, from left to right are shown the $p$-values, the $v$-values, and the policy.

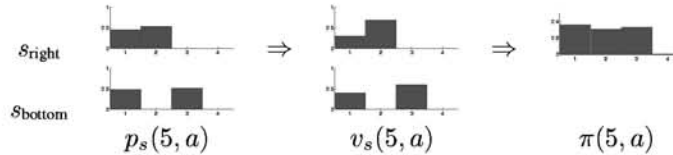

Figure 2: Transfer of the load-unload solution: plots of the same values as in figure 1 but with the left source absent. No additional learning was allowed (the left side plots are the same). The votes, however, change, and thus so does the final policy.

### 3.3 Nash Equilibrium Algorithm

To find a Nash Equilibrium, we start with $\alpha_s(x) = \beta_s(x)$ and $v_s(x, a) = p_s(x, a)$ and iterate to an equilibrium by repeatedly finding the best response for each source and simultaneously replacing the old solution with the new best responses. To prevent oscillation, whenever the change in $\alpha_s(x)v_s(x, a)$ grows from one step to the next, we replace the old solution with one halfway between the old and new solutions and continue the iteration.

## 4  Example Results

In all of these examples we used the same learning scheme. We ran the algorithm for a series of epochs. At each epoch, we calculated $\pi$ using the Nash Equilibrium algorithm. With probability $\epsilon$, we replace $\pi$ with one chosen uniformly over the simplex of conditional distributions. This insures some exploration. We follow $\pi$ for a fixed number of time steps and record the average reward for each source. We add these average rewards and the empirical estimate of the policy followed as data to the least-squares estimate of the returns. We then repeat for the next epoch.

### 4.1  Multiple Delivery Load-Unload Problem

We extend the classic load-unload problem to multiple receivers. The observation state is shown in figure 1. The hidden state is whether the agent is currently carrying cargo. Whenever the agent enters the top state (state 1), cargo is placed on the agent. Whenever the agent arrives in any of the boxed states while carrying cargo, the cargo is removed and the agent receives reward. For each boxed state, there is one reward source who only rewards for deliveries to that state (a reward of 1 for a delivery and 0 for all other time steps). In state 5, the agent has the choice of four actions each of which moves the agent to the corresponding state without error. Since the agent cannot observe neither whether it

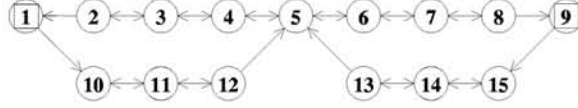

Figure 3: One-way door state diagram: At every state there are two actions (right and left) available to the agent. In states 1, 9, 10, and 15 where there are only single outgoing edges, both actions follow the same edge. With probability 0.1, an action will actually follow the other edge. Source 1 rewards entering state 1 whereas source 2 rewards entering state 9.

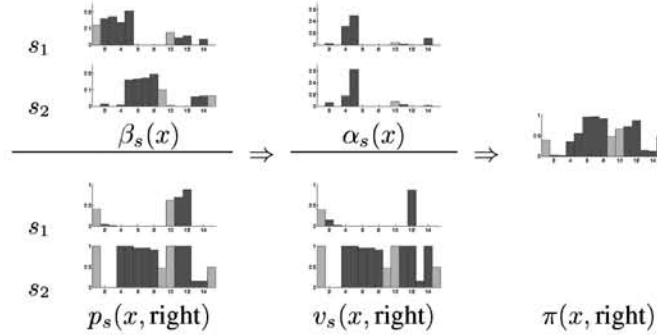

Figure 4: One-way door solution: from left to right: the sources' ideal policies, the votes, and the final agent's policy. Light bars are for states for which both actions lead to the same state.

has cargo nor its history, the optimal policy for state 5 is stochastic.

The algorithm set all $\alpha$- and $\beta$-values to 0 for states other than state 5. We started $\epsilon$ at 0.5 and reduced it to 0.1 by the end of the run. We ran for 300 epochs of 200 iterations by which point the algorithm consistently settled on the solution shown in figure 1. For each source, the algorithm found the best solution of randomly picking between the load state and the source's delivery state (as shown by the $p$-values). The votes are heavily weighted towards the delivery actions to overcome the other sources' preferences resulting in an approximately uniform policy. The important point is that, without additional learning, the policy can be changed if the left source leaves. The learned $\alpha$- and $p$-values are kept the same, but the Nash Equilibrium is different resulting in the policy in figure 2.

## 4.2 One-way Door Problem

In this case we consider the environment shown in figure 3. From each state the agent can move to the left or right except in states 1, 9, 10, and 15 where there is only one possible action. We can think of states 1 and 9 as one-way doors. Once the agent enters states 1 or 9, it may not pass back through except by going around through state 5. Source 1 gives reward when the agent passes through state 1. Source 2 gives reward when the agent passes through state 9. Actions fail (move in the opposite direction than intended) 0.1 of the time.

We ran the learning scheme for 1000 epochs of 100 iterations starting $\epsilon$ at 0.5 and reducing it to 0.015 by the last epoch. The algorithm consistently converged to the solution shown in figure 4. Source 1 considers the left-side states (2–5 and 11–12) the most important while source 2 considers the right-side states (5–8 and 13–14) the most important. The ideal policies captured by the $p$-values show that source 1 wants the agent to move left and source 2 wants the agent to move right for the upper states (2–8) while the sources

agree that for the lower states (11–14) the agent should move towards state 5. The votes reflect this preference and agreement. Both sources spend most of their vote on state 5, the state they both feel is important and on which they disagree. The other states (states for which only one source has a strong opinion or on which they agree), they do not need to spend much of their vote. The resulting policy is the natural one: in state 5, the agent randomly picks a direction after which, the agent moves around the chosen loop quickly to return to state 5. Just as in the load-unload problem, if we remove one source, the agent automatically adapts to the ideal policy for the remaining source (with only one source, $s_0$, present, $\pi(x, a) = p_{s_0}(x, a)$).

Estimating the optimal policies and then taking the mixture of these two policies would produce a far worse result. For states 2–8, both sources would have differing opinions and the mixture model would produce a uniform policy in those states; the agent would spend most of its time near state 5. Constructing a reward signal that is the sum of the sources' rewards does not lead to a good solution either. The agent will find that circling either the left or right loop is optimal and will have no incentive to ever travel along the other loop.

## 5  Conclusions

It is difficult to conceive of a method for providing a single reward signal that would result in the solution shown in figure 4 and still automatically change when one of the reward sources was removed. The biggest improvement in the algorithm will come from changing the form of the $\hat{R}_s^\pi$ estimator. For problems in which there is a single best solution, the KL-divergence measure seems to work well. However, we would like to be able to extend the load-unload result to the situation where the agent has a memory bit. In this case, the returns as a function of $\pi$ are bimodal (due to the symmetry in the interpretation of the bit). In general, allowing each source's preference to be modelled in a more complex manner could help extend these results.

### Acknowledgments

We would like to thank Charles Isbell, Tommi Jaakkola, Leslie Kaelbling, Michael Kearns, Satinder Singh, and Peter Stone for their discussions and comments.

This report describes research done within CBCL in the Department of Brain and Cognitive Sciences and in the AI Lab at MIT. This research is sponsored by a grants from ONR contracts Nos. N00014-93-1-3085 & N00014-95-1-0600, and NSF contracts Nos. IIS-9800032 & DMS-9872936. Additional support was provided by: AT&T, Central Research Institute of Electric Power Industry, Eastman Kodak Company, Daimler-Chrysler, Digital Equipment Corporation, Honda R&D Co., Ltd., NEC Fund, Nippon Telegraph & Telephone, and Siemens Corporate Research, Inc.

## References

[1] J. Hu and M. P. Wellman. Multiagent reinforcement learning: Theoretical framework and an algorithm. In *Proc. of the 15th International Conf. on Machine Learning*, pages 242–250, 1998.

[2] C. L. Isbell, C. R. Shelton, M. Kearns, S. Singh, and P. Stone. A social reinforcement learning agent. 2000. submitted to Autonomous Agents 2001.

[3] J. Karlsson. *Learning to Solve Multiple Goals*. PhD thesis, University of Rochester, 1997.

[4] M. Kearns, Y. Mansouor, and S. Singh. Fast planning in stochastic games. In *Proc. of the 16th Conference on Uncertainty in Artificial Intelligence*, 2000.

[5] M. L. Littman. Markov games as a framework for multi-agent reinforcement learning. In *Proc. of the 11th International Conference on Machine Learning*, pages 157–163, 1994.

[6] G. Owen. *Game Theory*. Academic Press, UK, 1995.

[7] S. Singh, M. Kearns, and Y. Mansour. Nash convergence of gradient dynamics in general-sum games. In *Proc. of the 16th Conference on Uncertainty in Artificial Intelligence*, 2000.

[8] S. P. Singh. The efficient learning of multiple task sequences. In *NIPS*, volume 4, 1992.
